# Asynchronous Distributed Learning of Topic Models

**Arthur Asuncion, Padhraic Smyth, Max Welling**
Department of Computer Science
University of California, Irvine
{asuncion,smyth,welling}@ics.uci.edu

## Abstract

Distributed learning is a problem of fundamental interest in machine learning and cognitive science. In this paper, we present asynchronous distributed learning algorithms for two well-known unsupervised learning frameworks: Latent Dirichlet Allocation (LDA) and Hierarchical Dirichlet Processes (HDP). In the proposed approach, the data are distributed across $P$ processors, and processors independently perform Gibbs sampling on their local data and communicate their information in a local asynchronous manner with other processors. We demonstrate that our asynchronous algorithms are able to learn global topic models that are statistically as accurate as those learned by the standard LDA and HDP samplers, but with significant improvements in computation time and memory. We show speedup results on a 730-million-word text corpus using 32 processors, and we provide perplexity results for up to 1500 virtual processors. As a stepping stone in the development of asynchronous HDP, a parallel HDP sampler is also introduced.

## 1 Introduction

Learning algorithms that can perform in a distributed asynchronous manner are of interest for several different reasons. The increasing availability of multi-processor and grid computing technology provides an immediate and practical motivation to develop learning algorithms that are able take advantage of such computational resources. Similarly, the increasing proliferation of networks of low-cost devices motivates the investigation of distributed learning in the context of sensor networks. On a deeper level, there are fundamental questions about distributed learning from the viewpoints of artificial intelligence and cognitive science.

In this paper, we focus on the specific problem of developing asynchronous distributed learning algorithms for a class of unsupervised learning techniques, specifically LDA [1] and HDP [2] with learning via Gibbs sampling. The frameworks of LDA and HDP have recently become popular due to their effectiveness at extracting low-dimensional representations from sparse high-dimensional data, with multiple applications in areas such as text analysis and computer vision. A promising approach to scaling these algorithms to large data sets is to distribute the data across multiple processors and develop appropriate distributed topic-modeling algorithms [3, 4, 5]. There are two somewhat distinct motivations for distributed computation in this context: (1) to address the memory issue when the original data and count matrices used by the algorithm exceed the main memory capacity of a single machine; and (2) using multiple processors to significantly speed up topic-learning, e.g., learning a topic model in near real-time for tens of thousands of documents returned by a search-engine.

While synchronous distributed algorithms for topic models have been proposed in earlier work, here we investigate *asynchronous* distributed learning of topic models. Asynchronous algorithms provide several computational advantages over their synchronous counterparts: (1) no global synchronization step is required; (2) the system is extremely fault-tolerant due to its decentralized nature; (3) heterogeneous machines with different processor speeds and memory capacities can be used; (4) new processors and new data can be incorporated into the system at any time.

Our primary novel contribution is the introduction of new asynchronous distributed algorithms for LDA and HDP, based on local collapsed Gibbs sampling on each processor. We assume an asyn-

chronous "gossip-based" framework [6] which only allows pairwise interactions between random processors. Our distributed framework can provide substantial memory and time savings over single-processor computation, since each processor only needs to store and perform Gibbs sweeps over $\frac{1}{P}$th of the data, where $P$ is the number of processors. Furthermore, the asynchronous approach can scale to large corpora and large numbers of processors, since no global synchronization steps are required. While building towards an asynchronous algorithm for HDP, we also introduce a novel synchronous distributed inference algorithm for HDP, again based on collapsed Gibbs sampling.

In the proposed framework, individual processors perform Gibbs sampling locally on each processor based on a noisy inexact view of the global topics. As a result, our algorithms are not necessarily sampling from the proper global posterior distribution. Nonetheless, as we will show in our experiments, these algorithms are empirically very robust and converge rapidly to high-quality solutions.

We first review collapsed Gibbs sampling for LDA and HDP. Then we describe the details of our distributed algorithms. We present perplexity and speedup results for our algorithms when applied to text data sets. We conclude with a discussion of related work and future extensions of our work.

## 2   A brief review of topic models

Before delving into the details of our distributed algorithms, we first describe the LDA and HDP topic models. In LDA, each document $j$ is modeled as a mixture over $K$ topics, and each topic $k$ is a multinomial distribution, $\phi_{wk}$, over a vocabulary of $W$ words[1]. Each document's mixture over topics, $\theta_{kj}$, is drawn from a Dirichlet distribution with parameter $\eta$. In order to generate a new document, $\theta_{kj}$ is first sampled from a Dirichlet distribution with parameter $\alpha$. For each token $i$ in that document, a topic assignment $z_{ij}$ is sampled from $\theta_{kj}$, and the specific word $x_{ij}$ is drawn from $\phi_{wz_{ij}}$. The graphical model for LDA is shown in Figure 1, and the generative process is below:

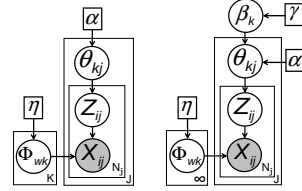

Figure 1:   Graphical models for LDA (left) and HDP (right).

$$\theta_{k,j} \sim D[\alpha] \qquad \phi_{w,k} \sim D[\eta] \qquad z_{ij} \sim \theta_{k,j} \qquad x_{ij} \sim \phi_{w,z_{ij}} \ .$$

Given observed data, it is possible to infer the posterior distribution of the latent variables. One can perform collapsed Gibbs sampling [7] by integrating out $\theta_{kj}$ and $\phi_{wk}$ and sampling the topic assignments in the following manner:

$$P(z_{ij} = k | z^{\neg ij}, w) \propto \frac{N_{wk}^{\neg ij} + \eta}{\sum_w N_{wk}^{\neg ij} + W\eta} \left( N_{jk}^{\neg ij} + \alpha \right) \ . \tag{1}$$

$N_{wk}$ denotes the number of word tokens of type $w$ assigned to topic $k$, while $N_{jk}$ denotes the number of tokens in document $j$ assigned to topic $k$. $N^{\neg ij}$ denotes the count with token $ij$ removed.

The HDP mixture model is composed of a hierarchy of Dirichlet processes. HDP is similar to LDA and can be viewed as the model that results from taking the infinite limit of the following finite mixture model. Let $L$ be the number of mixture components, and $\beta_k$ be top level Dirichlet variables drawn from a Dirichlet distribution with parameter $\gamma/L$. The mixture for each document, $\theta_{kj}$, is generated from a Dirichlet with parameter $\alpha\beta_k$. The multinomial topic distributions, $\phi_{wk}$ are drawn from a base Dirichlet distribution with parameter $\eta$. As in LDA, $z_{ij}$ is sampled from $\theta_{kj}$, and word $x_{ij}$ is sampled from $\phi_{wz_{ij}}$. If we take the limit of this model as $L$ goes to infinity, we obtain HDP:

$$\beta_k \sim D[\gamma/L] \qquad \theta_{k,j} \sim D[\alpha\beta_k] \qquad \phi_{w,k} \sim D[\eta] \qquad z_{ij} \sim \theta_{k,j} \qquad x_{ij} \sim \phi_{w,z_{ij}}.$$

To sample from the posterior, we follow the details of the direct assignment sampler for HDP [2]. Both $\theta_{kj}$ and $\phi_{wk}$ are integrated out, and $z_{ij}$ is sampled from a conditional distribution that is almost identical to that of LDA, except that a small amount of probability mass is reserved for the instantiation of a new topic. Note that although HDP is defined to have an infinite number of topics, the only topics that are instantiated are those that are actually used.

## 3   Asynchronous distributed learning for the LDA model

We consider the problem of learning an LDA model with $K$ topics in a distributed fashion where documents are distributed across $P$ processors. Each processor $p$ stores the following local variables:

$w_{ij}^p$ contains the word type for each token $i$ in document $j$ in the processor, and $z_{ij}^p$ contains the assigned topic for each token. $N_{wk}^{\neg p}$ is the global word-topic count matrix stored at the processor—this matrix stores counts of other processors gathered during the communication step and does not include the processor's local counts. $N_{kj}^p$ is the local document-topic count matrix (derived from $z^p$), $N_w^p$ is the simple word count on a processor (derived from $w^p$), and $N_{wk}^p$ is the local word-topic count matrix (derived from $z^p$ and $w^p$) which only contains the counts of data on the processor.

Newman et al. [5] introduced a parallel version of LDA based on collapsed Gibbs sampling (which we will call Parallel-LDA). In Parallel-LDA, each processor receives $\frac{1}{P}$ of the documents in the corpus and the $z$'s are globally initialized. Each iteration of the algorithm is composed of two steps: a Gibbs sampling step and a synchronization step. In the sampling step, each processor samples its local $z^p$ by using the global topics of the previous iteration. In the synchronization step, the local counts $N_{wk}^p$ on each processor are aggregated to produce a global set of word-topic counts $N_{wk}$. This process is repeated for either a fixed number of iterations or until the algorithm has converged.

Parallel-LDA can provide substantial memory and time savings. However, it is a fully synchronous algorithm since it requires global synchronization at each iteration. In some applications, a global synchronization step may not be feasible, e.g. some processors may be unavailable, while other processors may be in the middle of a long Gibbs sweep, due to differences in processor speeds. To gain the benefits of asynchronous computing, we introduce an asynchronous distributed version of LDA (Async-LDA) that follows a similar two-step process to that above. Each processor performs a local Gibbs sampling step followed by a step of communicating with another random processor.

For Async-LDA, during each iteration, the processors perform a full sweep of collapsed Gibbs sampling over their local topic assignment variables $z^p$ according to the following conditional distribution, in a manner directly analogous to Equation 1,

$$P(z_{pij} = k | z_p^{\neg ij}, w_p) \propto \frac{(N^{\neg p} + N^p)_{wk}^{\neg ij} + \eta}{\sum_w (N^{\neg p} + N^p)_{wk}^{\neg ij} + W\eta} \left( N_{pjk}^{\neg ij} + \alpha \right). \quad (2)$$

The combination of $N_{wk}^{\neg p}$ and $N_{wk}^p$ is used in the sampling equation. Recall that $N_{wk}^{\neg p}$ represents processor $p$'s belief of the counts of all the other processors with which it has already communicated (not including processor $p$'s local counts), while $N_{wk}^p$ is the processor's local word-topic counts. Thus, the sampling of the $z^p$'s is based on the processor's "noisy view" of the global set of topics.

Once the inference of $z^p$ is complete (and $N_{wk}^p$ is updated), the processor finds another finished processor and initiates communication[2]. We are generally interested in the case where memory and communication bandwidth are both limited. We also assume in the simplified gossip scheme that a processor can establish communication with every other processor – later in the paper we also discuss scenarios that relax these assumptions.

In the communication step, let us consider the case where two processors, $p$ and $g$ have never met before. In this case, processors simply exchange their local $N_{wk}^p$'s (their local contribution to the global topic set), and processor $p$ simply adds $N_{wk}^g$ to its $N_{wk}^{\neg p}$, and vice versa.

**Algorithm 1** Async-LDA

> **for** each processor $p$ in parallel **do**
>    **repeat**
>      Sample $z^p$ locally (Equation 2)
>      Receive $N_{wk}^g$ from random proc $g$
>      Send $N_{wk}^p$ to proc $g$
>      **if** $p$ has met $g$ before **then**
>        $N_{wk}^{\neg p} \leftarrow N_{wk}^{\neg p} - \tilde{N}_{wk}^g + N_{wk}^g$
>      **else**
>        $N_{wk}^{\neg p} \leftarrow N_{wk}^{\neg p} + N_{wk}^g$
>      **end if**
>    **until** convergence
> **end for**

Consider the case where two processors meet again. The processors should not simply swap and add their local counts again; rather, each processor should first remove from $N_{wk}^{\neg p}$ the previous influence of the other processor during their previous encounter, in order to prevent processors that frequently meet from over-influencing each other. We assume in the general case that a processor does not store in memory the previous counts of all the other processors that processor $p$ has already met. Since the previous local counts of the other processor were already absorbed into $N_{wk}^{\neg p}$ and are thus not retrievable, we must take a different approach. In Async-LDA, the processors exchange their $N_{wk}^p$'s, from which the count of words on each processor, $N_w^p$ can be derived. Using processor $g$'s $\tilde{N}_w^g$, processor $p$ creates $\tilde{N}_{wk}^g$ by sampling $N_w^g$ topic values randomly without replacement from

collection $\{N_{wk}^{\neg p}\}$. We can imagine that there are $\sum_k N_{wk}^{\neg p}$ colored balls, with $N_{wk}^{\neg p}$ balls of color $k$, from which we pick $N_w^g$ balls uniformly at random without replacement. This process is equivalent to sampling from a multivariate hypergeometric distribution. $\tilde{N}_{wk}^g$ acts as a substitute for the $N_{wk}^g$ that processor $p$ received during their previous encounter. Since all knowledge of the previous $N_{wk}^g$ is lost, this method can be justified by Laplace's principle of indifference (or the principle of maximum entropy). Finally, we update $N_{wk}^{\neg p}$ by subtracting $\tilde{N}_{wk}^g$ and adding the current $N_{wk}^g$:

$$N_{wk}^{\neg p} \leftarrow N_{wk}^{\neg p} - \tilde{N}_{wk}^g + N_{wk}^g \quad \text{where} \quad \tilde{N}_{w,k}^g \sim \text{MH}\left[N_w^g; N_{w,1}^{\neg p}, .., N_{w,K}^{\neg p}\right]. \quad (3)$$

Pseudocode for Async-LDA is provided in the display box for Algorithm 1. The assumption of limited memory can be relaxed by allowing processors to cache previous counts of other processors – the cached $N_{wk}^g$ would replace $\tilde{N}_{wk}^g$. We can also relax the assumption of limited bandwidth. Processor $p$ could forward its individual cached counts (from other processors) to $g$, and vice versa, to quicken the dissemination of information. In fixed topologies where the network is not fully connected, forwarding is necessary to propagate the counts across the network. Our approach can be applied to a wide variety of scenarios with varying memory, bandwidth, and topology constraints.

## 4 Synchronous and asynchronous distributed learning for the HDP model

Inference for HDP can be performed in a distributed manner as well. Before discussing our asynchronous HDP algorithm, we first describe a synchronous parallel inference algorithm for HDP.

We begin with necessary notation for HDPs: $\gamma$ is the concentration parameter for the top level Dirichlet Process (DP), $\alpha$ is the concentration parameter for the document level DP, $\beta_k$'s are top-level topic probabilities, and $\eta$ is the Dirichlet parameter for the base distribution. The graphical model for HDP is shown in Figure 1.

We introduce Parallel-HDP, which is analogous to Parallel-LDA except that new topics may be added during the Gibbs sweep. Documents are again distributed across the processors. Each processor maintains local $\beta_k^p$ parameters which are augmented when a new topic is locally created. During the Gibbs sampling step, each processor locally samples the $z^p$ topic assignments. In the synchronization step, the local word-topic counts $N_{wk}^p$ are aggregated into a single matrix of global counts $N_{wk}$, and the local $\beta_k^p$'s are averaged to form a global $\beta_k$. The $\alpha$, $\beta_k$ and $\gamma$ hyperparameters are also globally resampled during the synchronization step – see Teh et al. [2] for details. We fix $\eta$ to be a small constant. While $\alpha$ and $\gamma$ can also be fixed, sampling these parameters improves the rate of convergence. To facilitate sampling, relatively flat gamma priors are placed on $\alpha$ and $\gamma$. Finally, these parameters and the global count matrix are distributed back to the processors.

---

**Algorithm 2** Parallel-HDP

> **repeat**
>   **for** each processor $p$ in parallel **do**
>     Sample $z^p$ locally
>     Send $N_{wk}^p, \beta_k^p$ to master node
>   **end for**
>   $N_{wk} \leftarrow \sum_p N_{wk}^p$
>   $\beta_k \leftarrow \left(\sum_p \beta_k^p\right) / \text{P}$
>   Resample $\alpha, \beta_k, \gamma$ globally
>   Distribute $N_{wk}, \alpha, \beta_k, \gamma$ to all processors
> **until** convergence

**Algorithm 3** Async-HDP

> **for** each processor $p$ in parallel **do**
>   **repeat**
>     Sample $z^p$ and then $\alpha^p, \beta_k^p, \gamma^p$ locally
>     Receive $N_{wk}^g, \alpha^g, \beta_k^g$ from random proc $g$
>     Send $N_{wk}^p, \alpha^p, \beta_k^p$ to proc $g$
>     **if** $p$ has met $g$ before **then**
>       $N_{wk}^{\neg p} \leftarrow N_{wk}^{\neg p} - \tilde{N}_{wk}^g + N_{wk}^g$
>     **else**
>       $N_{wk}^{\neg p} \leftarrow N_{wk}^{\neg p} + N_{wk}^g$
>     **end if**
>     $\alpha^p \leftarrow (\alpha^p + \alpha^g) / 2$   and   $\beta_k^p \leftarrow (\beta_k^p + \beta_k^g) / 2$
>   **until** convergence
> **end for**

---

Motivated again by the advantages of local asynchronous communication between processors, we propose an Async-HDP algorithm. It is very similar in spirit to Async-LDA, and so we focus on the differences in our description. First, the sampling equation for $z^p$ is different to that of Async-LDA, since some probability mass is reserved for new topics:

$$P(z_{pij} = k | z_p^{\neg ij}, w_p) \propto \begin{cases} \frac{(N^{\neg p} + N^p)_{wk}^{\neg ij} + \eta}{\sum_w (N^{\neg p} + N^p)_{wk}^{\neg ij} + W\eta} \left(N_{pjk}^{\neg ij} + \alpha^p \beta_k^p\right), & \text{if } k \leq K_p \\[2ex] \frac{\alpha^p \beta_{\text{new}}^p}{W}, & \text{if } k \text{ is new.} \end{cases}$$

|                                            | KOS     | NIPS      | NYT        | PUBMED      |
|--------------------------------------------|---------|-----------|------------|-------------|
| Total number of documents in training set  | 3,000   | 1,500     | 300,000    | 8,200,000   |
| Size of vocabulary                         | 6,906   | 12,419    | 102,660    | 141,043     |
| Total number of words                      | 410,595 | 1,932,365 | 99,542,125 | 737,869,083 |
| Total number of documents in test set      | 430     | 184       | –          | –           |

Table 1: Data sets used for perplexity and speedup experiments

We resample the hyperparameters $\alpha^p$, $\beta_k^p$, $\gamma^p$ locally[3] during the inference step, and keep $\eta$ fixed.

In Async-HDP, a processor can add new topics to its collection during the inference step. Thus, when two processors communicate, the number of topics on each processor might be different. One way to merge topics is to perform bipartite matching across the two topic sets, using the Hungarian algorithm. However, performing this topic matching step imposes a computational penalty as the number of topics increases. In our experiments for Async-LDA, Parallel-HDP, and Async-HDP, we do not perform topic matching, but we simply combine the topics on different processors based their topic IDs and (somewhat surprisingly) the topics gradually self-organize and align. Newman et al. [5] also observed this same behavior occurring in Parallel-LDA.

During the communication step, the counts $N_{wk}^p$ and the parameters $\alpha^p$ and $\beta_k^p$ values are exchanged and merged. Async-HDP removes a processor's previous influence through the same MH technique used in Async-LDA. Pseudocode for Async-HDP is provided in the display box for Algorithm 3.

# 5   Experiments

We use four text data sets for evaluation: KOS, a data set derived from blog entries (dailykos.com); NIPS, a data set derived from NIPS papers (books.nips.cc); NYT, a collection of news articles from the New York Times (nytimes.com); and PUBMED, a large collection of PubMed abstracts (ncbi.nlm.nih.gov/pubmed/). The characteristics of these four data sets are summarized in Table 1.

For our perplexity experiments, parallel processors were simulated in software and run on smaller data sets (KOS, NIPS), to enable us to test the statistical limits of our algorithms. Actual parallel hardware is used to measure speedup on larger data sets (NYT, PUBMED). Our simulation features a gossip scheme over a fully connected network that lets each processor communicate with one other random processor at the end of every iteration, e.g., with $P=100$, there are 50 pairs at each iteration.

In our perplexity experiments, the data set is separated into a training set and a test set. We learn our models on the training set, and then we measure the performance of our algorithms on the test set using perplexity, a widely-used metric in the topic modeling community.

We briefly describe how perplexity is computed for our models. Perplexity is simply the exponentiated average per-word log-likelihood. For each of our experiments, we perform $S = 5$ different Gibbs runs, with each run lasting 1500 iterations (unless otherwise noted), and we obtain a sample at the end of each of those runs. The 5 samples are then averaged when computing perplexity. For Parallel-HDP, perplexity is calculated in the same way as in standard HDP:

$$\log p(\mathbf{x}^{\text{test}}) = \sum_{jw} \log \frac{1}{S} \sum_s \sum_k \hat{\theta}_{jk}^s \hat{\phi}_{wk}^s \text{ where } \hat{\theta}_{jk}^s = \frac{\alpha\beta_k + N_{jk}^s}{\sum_k (\alpha\beta_k) + N_j^s}, \ \hat{\phi}_{wk}^s = \frac{\eta + N_{wk}^s}{W\eta + N_k^s} \ . \quad (4)$$

After the model is run on the training data, $\hat{\phi}_{wk}^s$ is available in sample $s$. To obtain $\hat{\theta}_{jk}^s$, one must resample the topic assignments on the first half of each document in the test set while holding $\hat{\phi}_{wk}^s$ fixed. Perplexity is evaluated on the second half of each document in the test set, given $\hat{\phi}_{wk}^s$ and $\hat{\theta}_{jk}^s$.

The perplexity calculation for Async-LDA and Async-HDP uses the same formula. Since each processor effectively learns a separate local topic model, we can directly compute the perplexity for each processor's local model. In our experiments, we report the average perplexity among processors, and we show error bars denoting the minimum and maximum perplexity among all processors. The variance of perplexities between processors is usually quite small, which suggests that the local topic models learned on each processor are equally accurate.

For KOS and NIPS, we used the same settings for priors and hyperpriors: $\alpha = 0.1$, $\eta = 0.01$ for LDA and Async-LDA, and $\eta = 0.01$, $\gamma \sim \text{Gam}(10, 1)$, and $\alpha \sim \text{Gam}(2, 1)$ for the HDP algorithms.

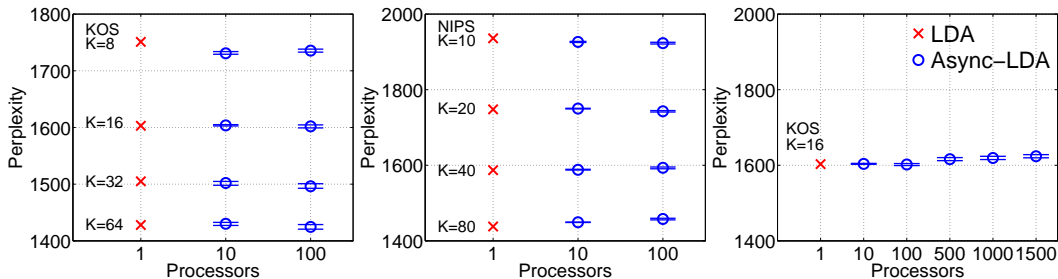

Figure 2: (a) Left: Async-LDA perplexities on KOS. (b) Middle: Async-LDA perplexities on NIPS. (c) Right: Async-LDA perplexities on KOS with many procs. Cache=5 when P≥100. 3000 iterations run when P≥500.

## 5.1 Async-LDA perplexity and speedup results

Figures 2(a,b) show the perplexities for Async-LDA on KOS and NIPS data sets for varying numbers of topics. The variation in perplexities between LDA and Async-LDA is slight and is significantly less than the variation in perplexities as the number of topics $K$ is changed. These numbers suggest that Async-LDA converges to solutions of the same quality as standard LDA. While these results are based on a single test/train split of the corpus, we have also performed cross-validation experiments (results not shown) which give essentially the same results across different test/train splits.

We also stretched the limits of our algorithm by increasing $P$ (e.g. for $P$=1500, there are only two documents on each processor), and we found that performance was virtually unchanged (figure 2(c)). As a baseline we ran an experiment where processors never communicate. As the number of processors $P$ was increased from 10 to 1500 the corresponding perplexities increased from 2600 to 5700, dramatically higher than our Async-LDA algorithm, indicating (unsurprisingly) that processor communication is essential to obtain good quality models. Figure 3(a) shows the rate of convergence of Async-LDA. As the number of processors increases, the rate of convergence slows, since it takes more iterations for information to propagate to all the processors. However, it is important to note that one iteration in real time of Async-LDA is up to $P$ times faster than one iteration of LDA. We show the same curve in terms of estimated real time in figure 3(b), assuming a parallel efficiency of 0.5, and one can see that Async-LDA converges much more quickly than LDA. Figure 3(c) shows actual speedup results for Async-LDA on NYT and PUBMED, and the speedups are competitive to those reported for Parallel-LDA [5]. As the data set size grows, the parallel efficiency increases, since communication overhead is dwarfed by the sampling time.

In Figure 3(a), we also show the performance of a baseline asynchronous averaging scheme, where global counts are averaged together: $N_{wk}^{\neg p} \leftarrow (N_{wk}^{\neg p} + N_{wk}^{\neg g})/d + N_{wk}^g$. To prevent unbounded count growth, $d$ must be greater than 2, and so we arbitrarily set $d$ to 2.5. While this averaging scheme initially converges quickly, it converges to a final solution that is worse than Async-LDA, regardless of the setting for $d$.

The rate of convergence for Async-LDA $P$=100 can be dramatically improved by letting each processor maintain a cache of previous $N_{wk}^g$ counts of other processors. Figures 3(a,b), $C$=5, show the improvement made by letting each processor cache the five most recently seen $N_{wk}^g$'s. Note that we still assume a limited bandwidth – processors do not forward individual cached counts, but instead share a single matrix of combined cache counts that helps the processors to achieve faster burn-in time. In this manner, one can elegantly make a tradeoff between time and memory.

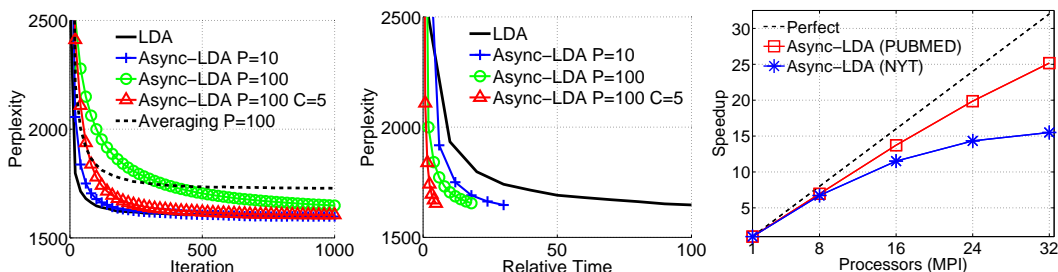

Figure 3: (a) Left: Convergence plot for Async-LDA on KOS, K=16. (b) Middle: Same plot with x-axis as relative time. (c) Right: Speedup results for NYT and PUBMED on a cluster, using Message Passing Interface.

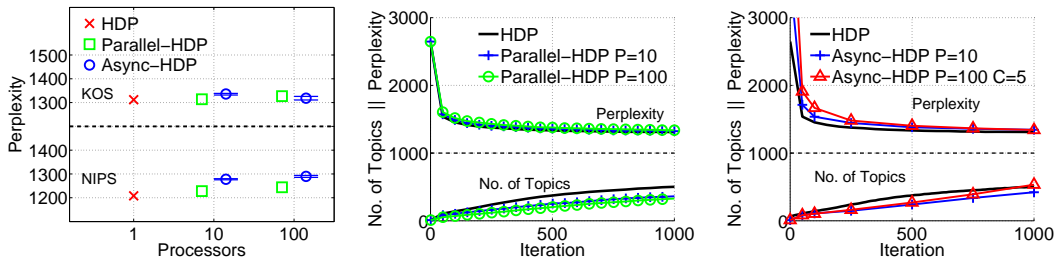

Figure 4: (a) Left: Perplexities for Parallel-HDP and Async-HDP. Cache=5 used for Async-HDP P=100. (b) Middle: Convergence plot for Parallel-HDP on KOS. (c) Right: Convergence plot for Async-HDP on KOS.

## 5.2 Parallel-HDP and Async-HDP results

Perplexities for Parallel-HDP after 1500 iterations are shown in figure 4(a), and they suggest that the model generated by Parallel-HDP has nearly the same predictive power as standard HDP. Figure 4(b) shows that Parallel-HDP converges at essentially the same rate as standard HDP on the KOS data set, even though topics are generated at a slower rate. Topics grow at a slower rate in Parallel-HDP since new topics that are generated locally on each processor are merged together during each synchronization step. In this experiment, while the number of topics is still growing, the perplexity has converged, because the newest topics are smaller and do not significantly affect the predictive power of the model. The number of topics does stabilize after thousands of iterations.

Perplexities for Async-HDP are shown in figures 4(a,c) as well. On the NIPS data set, there is a slight perplexity degradation, which is partially due to non-optimal parameter settings for $\alpha$ and $\gamma$. Topics are generated at a slightly faster rate for Async-HDP than for Parallel-HDP because Async-HDP take a less aggressive approach on pruning small topics, since processors need to be careful when pruning topics locally. Like Parallel-HDP, Async-HDP converges rapidly to a good solution.

## 5.3 Extended experiments for realistic scenarios

In certain applications, it is desirable to learn a topic model incrementally as new data arrives. In our framework, if new data arrives, we simply assign the new data to a new processor, and then let that new processor enter the "world" of processors with which it can begin to communicate. Our asynchronous approach requires no global initialization or global synchronization step. We do assume a fixed global vocabulary, but one can imagine schemes which allow the vocabulary to grow as well. We performed an experiment for Async-LDA where we introduced 10 new processors (each carrying new data) every 100 iterations. In the first 100 iterations, only 10% of the KOS data is known, and every 100 iterations, an additional 10% of the data is added to the system through new processors. Figure 5(a) shows that perplexity decreases as more processors and data are added. After 1000 iterations, the perplexity of Async-LDA has converged to the standard LDA perplexity. Thus, in this experiment, learning in an online fashion does not adversely affect the final model.

In the experiments previously described, documents were randomly distributed across processors. In reality, a processor may have a document set specialized to only a few topics. We investigated Async-LDA's behavior on a non-random distribution of documents over processors. After running LDA (K=20) on NIPS, we used the inferred mixtures $\theta_{jk}$ to separate the corpus into 20 different sets of documents corresponding to the 20 topics. We assigned 2 sets of documents to each of 10 processors, so that each processor had a document set that was specialized to 2 topics. Figure 5(b) shows that Async-LDA performs just as well on this non-random distribution of documents.

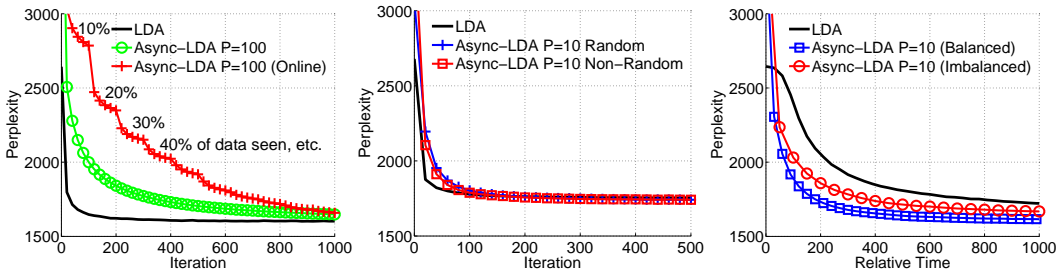

Figure 5: (a) Left: Online learning for Async-LDA on KOS, K=16. (b) Middle: Comparing random vs. non-random distribution of documents for Async-LDA on NIPS, K=20. (c) Right: Async-LDA on KOS, K=16, where processors have varying amounts of data. In all 3 cases, Async-LDA converges to a good solution.

Another situation of interest is the case where the amount of data on each processor varies. KOS was divided into 30 blocks of 100 documents and these blocks were assigned to 10 processors according to a distribution: $\{7, 6, 4, 3, 3, 2, 2, 1, 1, 1\}$. We assume that if a processor has $k$ blocks, then it will take $k$ units of time to complete one sampling sweep. Figure 5(c) shows that this load imbalance does not significantly affect the final perplexity achieved. More generally, the time $T_p$ that each processor $p$ takes to perform Gibbs sampling dictates the communication graph that will ensue. There exist pathological cases where the graph may be disconnected due to phase-locking (e.g. 5 processors with times $T = \{10, 12, 14, 19, 20\}$ where P1, P2, P3 enter the network at time 0 and P4, P5 enter the network at time 34). However, the graph is guaranteed to be connected over time if $T_p$ has a stochastic component (e.g. due to network delays), a reasonable assumption in practice.

In our experiments, we assumed a fully connected network of processors and did not focus on other network topologies. After running Async-LDA on both a 10x10 fixed grid network and a 100 node chain network on KOS $K$=16, we have verified that Async-LDA achieves the same perplexity as LDA as long as caching and forwarding of cached counts occurs between processors.

## 6 Discussion and conclusions

The work that is most closely related to that in this paper is that of Mimno and McCallum [3] and Newman et al. [5], who each propose parallel algorithms for the collapsed sampler for LDA. In other work, Nallapati et al. [4] parallelize the variational EM algorithm for LDA, and Wolfe et al. [8] examine asynchronous EM algorithms for LDA. The primary distinctions between our work and other work on distributed LDA based on Gibbs sampling are that (a) our algorithms use purely asynchronous communication rather than a global synchronous scheme, and (b) we have also extended these ideas (synchronous and asynchronous) to HDP. More generally, exact parallel Gibbs sampling is difficult to perform due to the sequential nature of MCMC. Brockwell [9] presents a pre-fetching parallel algorithm for MCMC, but this technique is not applicable to the collapsed sampler for LDA. There is also a large body of prior work on gossip algorithms (e.g., [6]), such as Newscast EM, a gossip algorithm for performing EM on Gaussian mixture learning [10].

Although processors perform local Gibbs sampling based on inexact global counts, our algorithms nonetheless produce solutions that are nearly the same as that of standard single-processor samplers. Providing a theoretical justification for these distributed algorithms is still an open area of research.

We have proposed a new set of algorithms for distributed learning of LDA and HDP models. Our perplexity and speedup results suggest that topic models can be learned in a scalable asynchronous fashion for a wide variety of situations. One can imagine our algorithms being performed by a large network of idle processors, in an effort to mine the terabytes of information available on the Internet.

**Acknowledgments**
This material is based upon work supported in part by NSF under Award IIS-0083489 (PS, AA), IIS-0447903 and IIS-0535278 (MW), and an NSF graduate fellowship (AA). MW was also supported by ONR under Grant 00014-06-1-073, and PS was also supported by a Google Research Award.

**References**

[1] D. Blei, A. Ng, and M. Jordan. Latent Dirichlet allocation. *JMLR*, 3:993–1022, 2003.

[2] Y. Teh, M. Jordan, M. Beal, and D. Blei. Hierarchical Dirichlet processes. *JASA*, 101(476), 2006.

[3] D. Mimno and A. McCallum. Organizing the OCA: learning faceted subjects from a library of digital books. In *JCDL '07*, pages 376–385, New York, NY, USA, 2007. ACM.

[4] R. Nallapati, W. Cohen, and J. Lafferty. Parallelized variational EM for latent Dirichlet allocation: An experimental evaluation of speed and scalability. In *ICDM Workshop On High Perf. Data Mining*, 2007.

[5] D. Newman, A. Asuncion, P. Smyth, and M. Welling. Distributed inference for latent Dirichlet allocation. In *NIPS 20*. MIT Press, Cambridge, MA, 2008.

[6] S. Boyd, A. Ghosh, B. Prabhakar, and D. Shah. Gossip algorithms: design, analysis and applications. In *INFOCOM*, pages 1653–1664, 2005.

[7] T. L. Griffiths and M. Steyvers. Finding scientific topics. *PNAS*, 101 Suppl 1:5228–5235, April 2004.

[8] J. Wolfe, A. Haghighi, and D. Klein. Fully distributed EM for very large datasets. In *ICML '08*, pages 1184–1191, New York, NY, USA, 2008. ACM.

[9] A. Brockwell. Parallel Markov chain Monte Carlo simulation by pre-fetching. *JCGS*, 15, No. 1, 2006.

[10] W. Kowalczyk and N. Vlassis. Newscast EM. In *NIPS 17*. MIT Press, Cambridge, MA, 2005.

## Footnotes

[1]To avoid clutter, we write $\phi_{wk}$ or $\theta_{kj}$ to denote the set of all components, i.e. $\{\phi_{wk}\}$ or $\{\theta_{kj}\}$. Similarly, when sampling from a Dirichlet, we write $\theta_{kj} \sim D[\alpha\beta_k]$ instead of $[\theta_{1,j}, ..\theta_{K,j}] \sim D[\alpha\beta_1, .., \alpha\beta_K]$.

[2] We don't discuss in general the details of how processors might identify other processors that have finished their iteration, but we imagine that a standard protocol could be used, like P2P.

[3]Sampling $\alpha^p$, $\beta_k^p$, $\gamma^p$ requires a global view of variables like $m_{\cdot k}$, the total number of "tables" serving "dish" $k$ [2]. These values can be asynchronously propagated in the same way that the counts are propagated.
